# On iterative Krylov-dogleg trust-region steps for solving neural networks nonlinear least squares problems

**Eiji Mizutani**
Department of Computer Science
National Tsing Hua University
Hsinchu, 30043 TAIWAN R.O.C.
*eiji@wayne.cs.nthu.edu.tw*

**James W. Demmel**
Mathematics and Computer Science
University of California at Berkeley,
Berkeley, CA 94720 USA
*demmel@cs.berkeley.edu*

## Abstract

This paper describes a method of dogleg trust-region steps, or restricted Levenberg-Marquardt steps, based on a projection process onto the Krylov subspaces for neural networks nonlinear least squares problems. In particular, the linear conjugate gradient (CG) method works as the *inner iterative* algorithm for solving the linearized Gauss-Newton normal equation, whereas the *outer* nonlinear algorithm repeatedly takes so-called "Krylov-dogleg" steps, relying only on matrix-vector multiplication without explicitly forming the Jacobian matrix or the Gauss-Newton model Hessian. That is, our *iterative* dogleg algorithm can reduce both operational counts and memory space by a factor of $O(n)$ (the number of parameters) in comparison with a *direct* linear-equation solver. This memory-less property is useful for large-scale problems.

## 1 Introduction

We consider the so-called **neural networks nonlinear least squares problem** [1] wherein the objective is to optimize the $n$ weight parameters of *neural networks* (NN) [e.g., *multilayer perceptrons* (MLP)], denoted by an $n$-dimensional vector $\boldsymbol{\theta}$, by minimizing the following:

$$E(\boldsymbol{\theta}) = \tfrac{1}{2}\sum_{p=1}^{m}(a_p(\boldsymbol{\theta}) - t_p)^2 = \tfrac{1}{2}\mathbf{r}^T(\boldsymbol{\theta})\mathbf{r}(\boldsymbol{\theta}), \qquad (1)$$

where $a_p(\boldsymbol{\theta})$ is the MLP output for the $p$th training data pattern and $t_p$ is the desired output. (Of course, these become vectors for a multiple-output MLP.) Here $\mathbf{r}(\boldsymbol{\theta})$ denotes the $m$-dimensional residual vector composed of $r_i(\boldsymbol{\theta})$, $i = 1, \ldots, m$, for all $m$ training data.

The gradient vector and Hessian matrix are given by $\mathbf{g} = \mathbf{g}(\boldsymbol{\theta}) \equiv \mathbf{J}^T\mathbf{r}$ and $\mathbf{H} = \mathbf{H}(\boldsymbol{\theta}) \equiv \mathbf{J}^T\mathbf{J}+\mathbf{S}$, where $\mathbf{J}$ is the $m \times n$ **Jacobian matrix** of $\mathbf{r}$, and $\mathbf{S}$ denotes the matrix of second-derivative terms. If $\mathbf{S}$ is simply omitted based on the "small residual" assumption, then the Hessian matrix reduces to the **Gauss-Newton model Hessian**: i.e., $\mathbf{J}^T\mathbf{J}$. Furthermore, a family of *quasi-Newton methods* can be applied to approximate term $\mathbf{S}$ alone, leading to the *augmented Gauss-Newton model Hessian* (see, for example, Mizutani [2] and references therein).

With any form of the aforementioned Hessian matrices, we can collectively write the following *Newton formula* to determine the next step $\boldsymbol{\delta}$ in the course of the **Newton iteration** for $\boldsymbol{\theta}_{\text{next}} = \boldsymbol{\theta}_{\text{now}} + \boldsymbol{\delta}$:

$$\mathbf{H}\boldsymbol{\delta} = -\mathbf{g}. \tag{2}$$

This linear system can be solved by a **direct solver** in conjunction with a suitable *matrix factorization*. However, typical criticisms towards the *direct algorithm* are:

- It is expensive to *form* and *solve* the linear equation (2), which requires $O(mn^2)$ operations when $m > n$;
- It is expensive to store the (symmetric) Hessian matrix $\mathbf{H}$, which requires $\frac{n(n+1)}{2}$ memory storage.

These issues may become much more serious for a *large-scale* problem.

In light of the vast literature on the nonlinear optimization, this paper describes how to alleviate these concerns, attempting to solve the Newton formula (2) approximately by **iterative** methods, which form a family of **inexact** (or **truncated**) **Newton methods** (see Dembo & Steihaug [3], for instance). An important subclass of the *inexact Newton* methods are *Newton-Krylov* methods. In particular, this paper focuses on a *Newton-CG*-type algorithm, wherein the *linear Gauss-Newton normal equation*,

$$(\mathbf{J}^T\mathbf{J})\boldsymbol{\delta} = -\mathbf{J}^T\mathbf{r}, \tag{3}$$

is solved iteratively by the **linear conjugate gradient** method (known as CGNR) for a **dogleg trust-region** implementation of the well-known *Levenberg-Marquardt* algorithm; hence, the name "dogleg trust-region Gauss-Newton-CGNR" algorithm, or "iterative Krylov-dogleg" method (similar to Steihaug [4]; Toint [5]).

## 2  Direct Dogleg Trust-Region Algorithms

In the NN literature, several variants of the Levenberg-Marquardt algorithm equipped with a *direct linear-equation solver*, particularly Marquardt's original method, have been recognized as instrumental and promising techniques; see, for example, Demuth & Beale [6]; Masters [7]; Shepherd [8]. They are based on a simple *direct* control of the Levenberg-Marquardt parameter $\mu$ in $(\mathbf{H}+\mu\mathbf{I})\boldsymbol{\delta} = -\mathbf{g}$, although such a simple $\mu$-control can cause a number of problems, because of a complicated relation between parameter $\mu$ and its associated step length (see Mizutani [9]).

Alternatively, a more efficient **dogleg** algorithm [10] can be employed that takes, depending on the size of trust region $R$, the Newton step $\boldsymbol{\delta}_{\text{Newton}}$ [i.e., the solution of Eq. (2)], the (restricted) Cauchy step $\boldsymbol{\delta}_{\text{Cauchy}}$, or an intermediate *dogleg step*:

$$\boldsymbol{\delta}_{\text{dogleg}} \stackrel{\text{def}}{=} \boldsymbol{\delta}_{\text{Cauchy}} + h(\boldsymbol{\delta}_{\text{Newton}} - \boldsymbol{\delta}_{\text{Cauchy}}), \tag{4}$$

which achieves a piecewise linear approximation to a trust-region step, or a restricted Levenberg-Marquardt step. Note that $\boldsymbol{\delta}_{\text{Cauchy}}$ is the step that minimizes the local

quadratic model in the steepest descent direction (i.e., Eq. (8) with $k = 1$). For details on Equation (4), refer to Powell [10]; Mizutani [9, 2].

When we consider the Gauss-Newton step for $\boldsymbol{\delta}_{\text{Newton}}$ in Equation (4), we must solve the overdetermined linear least squares problem: minimize$_{\boldsymbol{\delta}}$ $\|\mathbf{r} + \mathbf{J}\boldsymbol{\delta}\|_2$, for which three principal **direct linear-equation solvers** are:

(1) Normal equation approach (typically with Cholesky decomposition);
(2) QR decomposition approach to $\mathbf{J}\boldsymbol{\delta} = -\mathbf{r}$;
(3) Singular value decomposition (SVD) approach to $\mathbf{J}\boldsymbol{\delta} = -\mathbf{r}$ (only recommended when $\mathbf{J}$ is nearly rank-deficient).

Among those three *direct* solvers, approach (1) to Equation (3) is fastest. (For more details, refer to Demmel [11], Chapters 2 and 3.) In a *highly* overdetermined case (with a large data set; i.e., $m \gg n$), the dominant cost in approach (1) is the $mn^2$ operations to form the Gauss-Newton model Hessian by:

$$\mathbf{J}^T \mathbf{J} = \sum_{i=1}^{m} \mathbf{u}_i \mathbf{u}_i^T, \tag{5}$$

where $\mathbf{u}_i^T$ is the $i$th row vector of $\mathbf{J}$. This cost might be prohibitive even with enough storage for $\mathbf{J}^T \mathbf{J}$. Therefore, to overcome this limitation of direct solvers for Equation (3), we consider an iterative scheme in the next section.

## 3 Iterative Krylov-Dogleg Algorithm

The **iterative Krylov-dogleg** step approximates a trust-region step by iteratively approximating the Levenberg-Marquardt trajectory in the Krylov subspace via **linear conjugate gradient iterates** *until the approximate trajectory hits the trust-region boundary; i.e., a CG iterate falls outside the trust-region boundary*. In this context, the linear CGNR method is not intended to approximate the *full* Gauss-Newton step [i.e., the solution of Eq. (3)]. Therefore, the required number of CGNR-iterations might be kept small [see Section 4].

The iterative process for the linear-equation solution sequence $\{\boldsymbol{\delta}_k\}$ is called the *inner* [2] *iteration*, whereas the solution sequence $\{\boldsymbol{\theta}_k\}$ from the Krylov-dogleg algorithm is generated by the *outer iteration* (or *epoch*), as shown in Figure 1. We now describe the inner iteration algorithm, which is identical to the standard linear CG algorithm (see Demmel [11], pages 311-312) except steps 2, 4, and 5:

**Algorithm 3.1:** *The inner iteration of the Krylov-dogleg algorithm* (see Figure 1).

1. Initialization:
$$\boldsymbol{\delta}_0 = 0; \quad \mathbf{d}_0 = \mathbf{r}_0 = -\mathbf{g}_{\text{now}}, \text{ and } k = 1. \tag{6}$$
2. Matrix-vector product (compare Eq. (5) and see Algorithm 3.2):
$$\mathbf{z} = \mathbf{H}_{\text{now}} \mathbf{d}_k = \mathbf{J}_{\text{now}}^T (\mathbf{J}_{\text{now}} \mathbf{d}_k) = \sum_{i=1}^{m} (\mathbf{u}_i^T \mathbf{d}_k) \mathbf{u}_i. \tag{7}$$

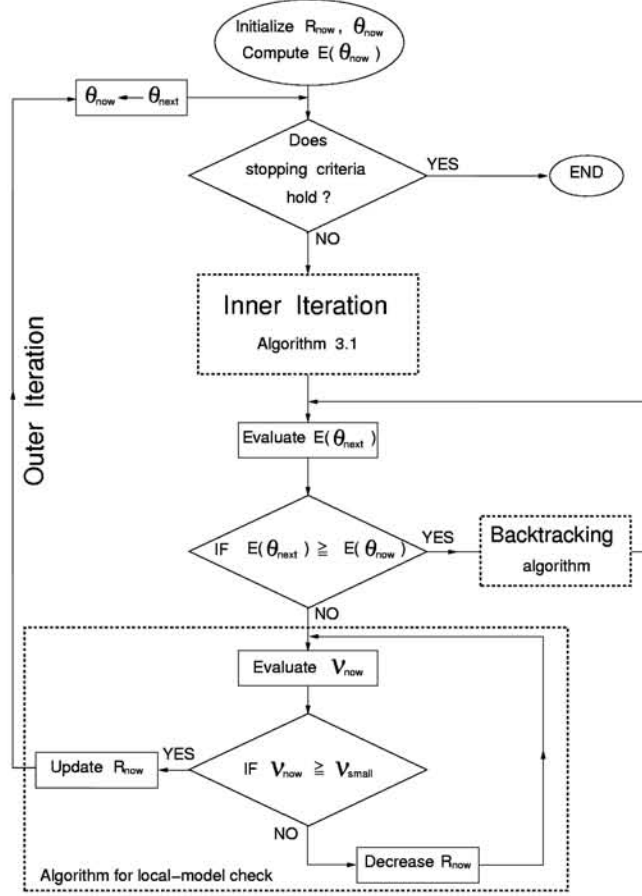

Figure 1: *The algorithmic flow of an iterative Krylov-dogleg algorithm. For detailed procedures in the three dotted rectangular boxes, refer to Mizutani and Demmel [12] and Algorithm 3.1 in text.*

3. Analytical step size: $\eta_k = \dfrac{\mathbf{r}_{k-1}^T \mathbf{r}_{k-1}}{\mathbf{d}_k^T \mathbf{z}}$.

4. Approximate solution:
$$\boldsymbol{\delta}_k = \boldsymbol{\delta}_{k-1} + \eta_k \mathbf{d}_k. \tag{8}$$

   If $\|\boldsymbol{\delta}_k\| < R_{\mathrm{now}}$, then go onto the next step 5; otherwise compute
$$\boldsymbol{\delta}_k = R_{\mathrm{now}} \frac{\boldsymbol{\delta}_k}{\|\boldsymbol{\delta}_k\|}, \tag{9}$$

   and **terminate**.

5. Linear-system residual: $\mathbf{r}_k = \mathbf{r}_{k-1} - \eta_k \mathbf{z}$.

   If $\|\mathbf{r}_k\|_2$ is small enough, then set $R_{\mathrm{now}} \leftarrow \|\boldsymbol{\delta}_k\|$, and **terminate**.

   Otherwise, continue with step 6.

6. Improvement: $\beta_{k+1} = \dfrac{\mathbf{r}_k^T \mathbf{r}_k}{\mathbf{r}_{k-1}^T \mathbf{r}_{k-1}}$.

7. Search direction: $\mathbf{d}_{k+1} = \mathbf{r}_k + \beta_{k+1} \mathbf{d}_k$. Then, set $k = k + 1$ and back to step 2.

The first step given by Equation (8) is always the Cauchy step $\boldsymbol{\delta}_{\text{Cauchy}}$, moving $\boldsymbol{\theta}_{\text{now}}$ to the Cauchy point $\boldsymbol{\theta}_{\text{Cauchy}}$ when $R_{\text{now}} > \|\boldsymbol{\delta}_{\text{Cauchy}}\|$. Then, departing from $\boldsymbol{\theta}_{\text{Cauchy}}$, the linear CG constructs a Krylov-dogleg trajectory (by adding a CG point one by one) towards the Gauss-Newton point $\boldsymbol{\theta}_{\text{Newton}}$ until the constructed trajectory hits the trust-region boundary (i.e., $\|\boldsymbol{\delta}_k\| \geq R_{\text{now}}$ is satisfied in step 4), or till the linear-system residual becomes small in step 5 (unlikely to occur for small forcing terms; e.g., 0.01). In this way, the algorithm computes a vector between the steepest descent direction and the Gauss-Newton direction, resulting in an approximate Levenberg-Marquardt step in the Krylov subspace.

In step 2, the matrix-vector multiplication of $\mathbf{H}\mathbf{d}_k$ in Equation (7) can be performed with neither the Jacobian nor Hessian matrices explicitly required, keeping only several $n$-dimensional vectors in memory at the same time, as shown next:

**Algorithm 3.2**: *Matrix-vector multiplication step.*

*for $i = 1$ to $m$; i.e., one sweep of all training data:*

      (a) do forward propagation to compute the MLP output $a_i(\boldsymbol{\theta})$ for datum $i$;

      (b) do backpropagation [3] to obtain the $i$th row vector $\mathbf{u}_i^T$ of matrix $\mathbf{J}$;

      (c) compute $(\mathbf{u}_i^T \mathbf{d}_k)\mathbf{u}_i$ and add it to $\mathbf{z}$;

*end for.*

For one sweep of all $m$ data, each of steps (a) and (b) costs at least $2mn$ (plus additional costs that depend on the MLP architectures) and step (c) [i.e., Eq. (7)] costs $4mn$. Hence, the overall cost of the inner iteration (Algorithm 3.1) can be kept as $O(mn)$, especially when the number of inner iterations is small owing to our strategy of upper-bounded trust-region radii (e.g., $R_{upper} = 1$ for the parity problem). Note for "Algorithm for local-model check" in Figure 1 that evaluating $\nu_{\text{now}}$ (a ratio between the actual error reduction and the reduction predicted by the current local quadratic model) needs a procedure similar to Algorithm 3.2. For more details on the algorithm in Figure 1, refer to Mizutani and Demmel [12].

## 4  Experiments and Discussions

In the NN literature, there are numerous algorithmic comparisons available (see, for example, Moller [14]; Demuth & Beale [6]; Shepherd [8]; Mizutani [2, 9, 16]). Due to the space limitation, this section compares typical behaviors of our Krylov-dogleg Gauss-Newton CGNR (or iterative dogleg) algorithm and Powell's dogleg-based algorithm with a direct linear-equation solver (or direct dogleg) for solving highly *overdetermined* parity problems. In our numerical tests, we used a criterion, in which the MLP output for the $p$th pattern, $a_p$, can be regarded as either "on" (1.0) if $a_p \geq 0.8$, or "off" (-1.0) if $a_p \leq -0.8$; otherwise, it is "undecided." The initial parameter set was randomly generated in the range $[-0.3, 0.3]$, and the two algorithms started exactly at the same point in the parameter space.

Figure 2 presents MLP-learning curves in RMSE (root mean squared error) for the 20-bit and 14-bit parity problems. In (b) and (c), the total execution time [roughly (b) 32 days (500 epochs); (c) two hours (450 epochs), both on 299-MHz UltraSparc] of the direct dogleg algorithm was normalized for comparison purpose. Notably, the

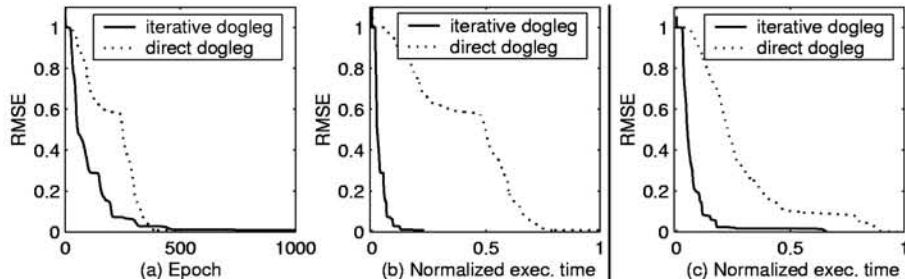

Figure 2: *MLP-learning curves of RMSE (root mean squared error) obtained by the "iterative dogleg" (solid line) and the "direct dogleg" (broken line): (a) "epoch" and (b) "normalized execution time" for the 20-bit parity problem with a standard $20 \times 19 \times 1$ MLP with hyperbolic tangent node functions ($m = 2^{20}$, $n = 419$), and (c) "normalized execution time" for the 14-bit parity problem with a $14 \times 13 \times 1$ MLP ($m = 2^{14}$, $n = 209$). In (a),(b), the iterative dogleg reduced the number of incorrect patterns down to 21 (nearly RMSE = 0.009) at epoch 838, whereas the direct dogleg reached the same error level at epoch 388. In (c), the iterative dogleg solved it perfectly at epoch 1,034 and the direct dogleg did so at epoch 401.*

iterative dogleg converged faster to a small RMSE [4] than the direct dogleg at an early stage of learning even with respect to epoch. Moreover, the average number of inner CG iterations per epoch in the iterative dogleg algorithm was quite small, 5.53 for (b) and 4.61 for (c). Thus, the iterative dogleg worked nearly (b) nine times and (c) four times faster than the direct dogleg in terms of the average execution time per epoch. Those speed-up ratios became smaller than $n$ mainly due to the aforementioned cost of Algorithm 3.2. Yet, as $n$ increases, the speed-up ratio can be larger especially when the number of inner iterations is reasonably small.

## 5   Conclusion and Future Directions

We have compared two batch-mode MLP-learning algorithms: *iterative* and *direct* dogleg trust-region algorithms. Although such a high-dimensional parity problem is very special in the sense that it involves a large data set but the size of MLP can be kept relatively small, the algorithmic features of the two dogleg methods can be well understood from the obtained experimental results. That is, the iterative dogleg has the great advantage of reducing the cost of an epoch from $O(mn^2)$ to $O(mn)$, and the memory requirements from $O(n^2)$ to $O(n)$, a factor of $O(n)$ in both cases. When $n$ is large, this is a very large improvement. It also has the advantage of faster convergence in the early epochs, achieving a lower RMSE after fewer epochs than the direct dogleg. Its disadvantage is that it may need more epochs to converge to a very small RMSE than the direct dogleg (although it might work faster in execution time). Thus, the iterative dogleg is most attractive when attempting to achieve a reasonably small RMSE on very large problems in a short period of time.

The iterative dogleg is a matrix-free algorithm that extracts information about the Hessian matrix via matrix-vector multiplication; this algorithm might be characterized as **iterative batch-mode learning**, an intermediate between *direct* batch-

mode learning and online pattern-by-pattern learning. Furthermore, the algorithm might be implemented in a block-by-block updating mode if a large data set can be split into multiple proper-size data blocks; so, it would be of our great interest to compare the performance with online-mode learning algorithms for solving large-scale real-world problems with a large-scale NN model.

**Acknowledgments**

We would like to thank Stuart Dreyfus (IEOR, UC Berkeley) and Rich Vuduc (CS, UC Berkeley) for their valuable advice. The work was supported in part by SONY US Research Labs., and in part by "Program for Promoting Academic Excellence of Universities," grant 89-E-FA04-1-4, Ministry of Education, Taiwan.

## Footnotes

[1]The posed problem can be viewed as an *implicitly constrained optimization* problem as long as hidden-node outputs are produced by sigmoidal "squashing" functions [1]. Our algorithm exploits the special structure of the sum of squared error measure in Equation (1); hence, the other objective functions are outside the scope of this paper.

[2]**Nonlinear conjugate gradient** methods, such as Polak-Ribiere's CG (see Mizutani and Jang [13]) and Moller's scaled CG [14], are also widely-employed for training MLPs, but those *nonlinear* versions attempt to approximate the entire Hessian matrix by generating the solution sequence $\{\boldsymbol{\theta}_k\}$ directly as the *outer* nonlinear algorithm. Thus, they ignore the special structure of the nonlinear least squares problem; so does Pearlmutter's method [15] to the Newton formula, although its modification may be possible.

[3]The batch-mode MLP backpropagation can be viewed as an efficient matrix-vector multiplication ($2mn$ operations) for computing the gradient $\mathbf{J}^T\mathbf{r}$ *without forming explicitly* the $m \times n$ Jacobian matrix or the $m$-dimensional residual vector (with some extra costs).

[4] A standard steepest descent-type online pattern-by-pattern learning (or incremental gradient) algorithm (with or without a momentum term) failed to converge to a small RMSE in those parity problems due to *hidden-node saturation* [1].

# References

[1] E. Mizutani, S. E. Dreyfus, and J.-S. R. Jang. On dynamic programming-like recursive gradient formula for alleviating hidden-node satuaration in the parity problem. In *Proceedings of the International Workshop on Intelligent Systems Resolutions – the 8th Bellman Continuum*, pages 100–104, Hsinchu, TAIWAN, 2000.

[2] Eiji Mizutani. Powell's dogleg trust-region steps with the quasi-Newton augmented Hessian for neural nonlinear least-squares learning. In *Proceedings of the IEEE Int'l Conf. on Neural Networks (vol.2)*, pages 1239–1244, Washington, D.C., July 1999.

[3] R. S. Dembo and T. Steihaug. Truncated-Newton algorithms for large-scale unconstrained optimization. *Math. Prog.*, 26:190–212, 1983.

[4] Trond Steihaug. The conjugate gradient method and trust regions in large scale optimization. *SIAM J. Numer. Anal.*, 20(3):626–637, 1983.

[5] P. L. Toint. On large scale nonlinear least squares calculations. *SIAM J. Sci. Statist. Comput.*, 8(3):416–435, 1987.

[6] H. Demuth and M. Beale. *Neural Network Toolbox for Use with MATLAB*. The MathWorks, Inc., Natick, Massachusetts, 1998. User's Guide (version 3.0).

[7] Timothy Masters. *Advanced algorithms for neural networks: a C++ sourcebook*. John Wiley & Sons, New York, 1995.

[8] Adrian J. Shepherd. *Second-Order Methods for Neural Networks: Fast and Reliable Training Methods for Multi-Layer Perceptrons*. Springer-Verlag, 1997.

[9] Eiji Mizutani. Computing Powell's dogleg steps for solving adaptive networks nonlinear least-squares problems. In *Proc. of the 8th Int'l Fuzzy Systems Association World Congress (IFSA'99), vol.2*, pages 959–963, Hsinchu, Taiwan, August 1999.

[10] M. J. D. Powell. A new algorithm for unconstrained optimization. In *Nonlinear Programming*, pages 31–65. Edited by J.B. Rosen et al., Academic Press, 1970.

[11] James W. Demmel. *Applied Numerical Linear Algebra*. SIAM, 1997.

[12] Eiji Mizutani and James W. Demmel. On generalized dogleg trust-region steps using the Krylov subspace for solving neural networks nonlinear least squares problems. Technical report, Computer Science Dept., UC Berkeley, 2001. (In preparation).

[13] E. Mizutani and J.-S. R. Jang. Chapter 6: Derivative-based Optimization. In *Neuro-Fuzzy and Soft Computing*, pages 129–172. J.-S. R. Jang, C.-T. Sun and E. Mizutani. Prentice Hall, 1997.

[14] Martin Fodslette Moller. A scaled conjugate gradient algorithm for fast supervised learning. *Neural Networks*, 6:525–533, 1993.

[15] B. A. Pearlmutter. Fast exact multiplication by the Hessian. *Neural Computation*, 6(1):147–160, 1994.

[16] E. Mizutani, K. Nishio, N. Katoh, and M. Blasgen. Color device characterization of electronic cameras by solving adaptive networks nonlinear least squares problems. In *Proc. of the 8th IEEE Int'l Conf. on Fuzzy Systems, vol. 2*, pages 858–862, 1999.
